# Convex Learning with Invariances

**Choon Hui Teo**
Australian National University
choonhui.teo@anu.edu.au

**Amir Globerson**
CSAIL, MIT
gamir@csail.mit.edu

**Sam Roweis**
Department of Computer Science
University of Toronto
roweis@cs.toronto.edu

**Alexander J. Smola**
NICTA
Canberra, Australia
alex.smola@gmail.com

## Abstract

Incorporating invariances into a learning algorithm is a common problem in machine learning. We provide a convex formulation which can deal with arbitrary loss functions and arbitrary losses. In addition, it is a drop-in replacement for most optimization algorithms for kernels, including solvers of the SVMStruct family. The advantage of our setting is that it relies on column generation instead of modifying the underlying optimization problem directly.

## 1 Introduction

Invariances are one of the most powerful forms of prior knowledge in machine learning; they have a long history [9, 1] and their application has been associated with some of the major success stories in pattern recognition. For instance, the insight that in vision tasks, one should be often be designing detectors that are invariant with respect to translation, small degrees of rotation & scaling, and image intensity has led to best-in-class algorithms including tangent-distance [13], virtual support vectors [5] and others [6].

In recent years a number of authors have attempted to put learning with invariances on a solid mathematical footing. For instance, [3] discusses how to extract invariant features for estimation and learning *globally* invariant estimators for a known class of invariance transforms (preferably arising from Lie groups). Another mathematically appealing formulation of the problem of learning with invariances casts it as a second order cone programming [8]; unfortunately this is neither particularly efficient to implement (having worse than cubic scaling behavior) nor does it cover a wide range of invariances in an automatic fashion. A different approach has been to pursue "robust" estimation methods which, roughly speaking, aim to find estimators whose performance does not suffer significantly when the observed inputs are degraded in some way. Robust estimation has been applied to learning problems in the context of missing data [2] and to deal with specific type of data corruption at test time [7]. The former approach again leads to a second order cone program, limiting its applicability to very small datasets; the latter is also computationally demanding and is limited to only specific types of data corruption.

Our goal in this work is to develop a computationally scalable and broadly applicable approach to supervised learning with invariances which is easily adapted to new types of problems and can take advantage of existing optimization infrastructures. In this paper we propose a method which has what we believe are many appealing properties:

1. It formulates invariant learning as a convex problem and thus can be implemented directly using any existing convex solver, requiring minimal additional memory and inheriting the convergence properties/guarantees of the underlying implementation.

2. It can deal with *arbitrary* invariances, including *gradual degradations*, provided that the user provides a computational recipe to generate invariant equivalents efficiently from a given data vector.
3. It provides a unifying framework for a number of previous approaches, such as the method of Virtual Support Vectors [5] and is broadly applicable not just to binary classification but in fact to any structured estimation problem in the sense of [16].

## 2  Maximum Margin Loss with Invariances

We begin by describing a maximum margin formulation of supervised learning which naturally incorporates invariance transformations on the input objects. We assume that we are given input patterns $x \in \mathcal{X}$ from from some space $\mathcal{X}$ and that we want to estimate outputs $y \in \mathcal{Y}$. For instance $\mathcal{Y} = \{\pm 1\}$ corresponds to binary classification; $\mathcal{Y} = A^n$ corresponds to sequence prediction over the alphabet $A$.[1] We denote our prediction by $\bar{y}(x)$, which is obtained by maximizing our learned function $f : \mathcal{X} \times \mathcal{Y} \to \mathbb{R}$, i.e. $\bar{y}(x) := \operatorname{argmax}_{y \in \mathcal{Y}} f(x, y)$. For instance, if we are training a (generative or discriminative) probabilistic model, $f(x, y) = \log p(y|x)$ then our prediction is the maximum a-posteriori estimate of the target $y$ given $x$. In many interesting cases $\bar{y}(x)$ is obtained by solving a nontrivial discrete optimization problem, e.g. by means of dynamic programming. In kernel methods $f(x, y) = \langle \phi(x, y), w \rangle$ for a suitable feature map $\phi$ and weight vector $w$. For the purpose of our analysis the precise form of $f$ is immaterial, although our experiments focus on the kernel machines, due to the availability of scalable optimizers for that class of estimators.

### 2.1  Invariance Transformations and Invariance Sensitive Cost

The crucial ingredient to formulating invariant learning is to capture the domain knowledge that there exists some class $\mathcal{S}$ of invariance transforms $s$ which can act on the input $x$ while leaving the target $y$ essentially unchanged. We denote by $(s(x), y)\ s \in \mathcal{S}$ the set of valid transformations of the pair $(x, y)$. For instance, we might believe that slight rotation (in pixel coordinates) of an input image in a pattern recognition problem do not change the image label. For text classification problems such as spam filtering, we may believe that certain editing operations (such as changes in capitalization or substitutions like Viagra $\to$ Vlagra,V!agra) should not affect our decision function. Of course, most invariances only apply "locally", i.e. in the neighborhood of the original input vector. For instance, rotating an image of the digit 6 too far might change its label to 9; applying both a substitution and an insertion can change Viagra $\to$ diagram. Furthermore, certain invariances may only hold for certain *pairs* of input and target. For example, we might believe that horizontal reflection is a valid invariance for images of digits in classes 0 and 8 but not for digits in class 2. The set $s(x)\ s \in \mathcal{S}$ incorporates both the locality and applicability constraints. (We have introduced a slight abuse of notation since $s$ may depend on $y$ but this should always be clear in context.)

To complete the setup, we adopt the standard assumption that the world or task imposes a cost function such that if the true target for an input $x$ is $y$ and our prediction is $\bar{y}(x)$ we suffer a cost $\Delta(y, \bar{y}(x))$.[2] For learning with invariances, we extend the definition of $\Delta$ to include the invariance function $s(x)$, if any, which was applied to the input object: $\Delta(y, \bar{y}(s(x)), s)$. This allows the cost to depend on the transformation, for instance we might suffer less cost for poor predictions when the input has undergone very extreme transformations. In a image labeling problem, for example, we might believe that a lighting/exposure invariance applies but we might want to charge small cost for extremely over-exposed or under-exposed images since they are almost impossible to label. Similarly, we might assert that scale invariance holds but give small cost to severely spatially down-sampled images since they contain very little information.

### 2.2  Max Margin Invariant Loss

Our approach to the invariant learning problem is very natural, yet allows us to make a surprising amount of analytical and algorithmic progress. A key quantity is the cost under the *worst case transformation* for each example, i.e. the transformation under which our predicted target suffers

the maximal cost compared with the true target:

$$C(x, y, f) = \sup_{s \in \mathcal{S}} \Delta(y, \bar{y}(s(x)), s) \tag{1}$$

The objective function (loss) that we advocate minimizing during learning is essentially a convex upper bound on this worst case cost which incorporates a notion of (scaled) margin:

$$l(x, y, f) := \sup_{y' \in \mathcal{Y}, s \in \mathcal{S}} \Gamma(y, y')(f(s(x), y') - f(s(x), y)) + \Delta(y, y', s) \tag{2}$$

This loss function finds the combination of invariance transformation and predicted target for which the sum of (scaled) "margin violation" plus the cost is maximized. The function $\Gamma(y, y')$ is a non-negative margin scaling which allows different target/prediction pairs to impose different amounts of loss on the final objective function.[3] The numerical scale of $\Gamma$ also sets the regularization tradeoff between margin violations and the prediction cost $\Delta$.

This loss function has two mathematically important properties which allow us to develop scalable and convergent algorithms as proposed above.

**Lemma 1** *The loss $l(x, y, f)$ is convex in $f$ for any choice of $\Gamma$, $\Delta$ and $\mathcal{S}$.*

**Proof** For fixed $(y', s)$ the expression $\Gamma(y, y')(f(s(x), y') - f(s(x), y)) + \Delta(y, y', s)$ is linear in $f$, hence (weakly) convex. Taking the supremum over a set of convex functions yields a convex function. ∎

This means that we can plug $l$ into any convex solver, in particular whenever $f$ belongs to a linear function class, as is the case with kernel methods. The primal (sub)gradient of $l$ is easy to write:

$$\partial_f l(x, y, f) = \Gamma(y, y^*)(\phi(s^*(x), y^*) - \phi(s^*(x), y)) \tag{3}$$

where $s^*, y^*$ are values of $s, y$ for which the supremum in Eq. (2) is attained and $\phi$ is the evaluation functional of $f$, that is $\langle f, \phi(x, y) \rangle = f(x, y)$. In kernel methods $\phi$ is commonly referred to as the *feature map* with associated kernel

$$k((x, y), (x', y')) = \langle \phi(x, y), \phi(x', y') \rangle. \tag{4}$$

Note that there is no need to define $\mathcal{S}$ formally. All we need is a computational recipe to obtain the worst case $s \in \mathcal{S}$ in terms of the scaled margin in Eq. 2. Nor is there any requirement for $\Delta(y, y', s)$ or $(s(x), y)$ to have any particularly appealing mathematical form, such as the polynomial trajectory required by [8], or the ellipsoidal shape described by [2].

**Lemma 2** *The loss $l(x, y, f)$ provides an upper bound on $C(x, y, f) = \sup_{s \in \mathcal{S}} \Delta(y, \bar{y}(s(x)), s)$.*

**Proof** Denote by $(s^*, y^*)$ the values for which the supremum of $C(x, y, f)$ is attained. By construction $f(s^*(x), y^*) \geq f(s^*(x), y)$. Plugging this inequality into Eq. (2) yields

$$l(x, y, f) \geq \Gamma(y, y^*)(f(s^*(x), y^*) - f(s^*(x), y)) + \Delta(y, y^*, s^*) \geq \Delta(y, y^*, s^*).$$

Here the first inequality follows by substituting $(s^*, y^*)$ into the supremum. The second inequality follows from the fact that $\Gamma \geq 0$ and that $(s^*, y^*)$ are the maximizers of the empirical loss. ∎

This is essentially a direct extension of [16]. The main modifications are the inclusion of a margin scale $\Gamma$ and the use of an invariance transform $s(x)$. In section 4 we clarify how a number of existing methods for dealing with invariances can be viewed as special cases of Eq. (2).

In summary, Eq. (2) penalizes estimation errors not only for the observed pair $(x, y)$ but also for patterns $s(x)$ which are "near" $x$ in terms of the invariance transform $s$. Recall, however, that the cost function $\Delta$ may assign quite a small cost to a transformation $s$ which takes $x$ very far away from the original. Furthermore, the transformation class is restricted only by the computational consideration that we can efficiently find the "worst case" transformation; $\mathcal{S}$ does not have to have a specific analytic form. Finally, there is no specific restriction on $y$, thus making the formalism applicable to any type of structured estimation.

# 3 Learning Algorithms for Minimizing Invariant Loss

We now turn to the question of learning algorithms for our invariant loss function. We assume that we are given a training set of input patterns $X = \{x_1, \ldots, x_m\}$ and associated labels $Y = \{y_1, \ldots, y_m\}$. We follow the common approach of minimizing, at training time, our average training loss plus a penalty for model complexity. In the context of kernel methods this can be viewed as a regularized empirical risk functional of the form

$$R[f] = \frac{1}{m} \sum_{i=1}^{m} l(x_i, y_i, f) + \frac{\lambda}{2} \|f\|_{\mathcal{H}}^2 \text{ where } f(x, y) = \langle \phi(x, y), w \rangle. \tag{5}$$

A direct extension of the derivation of [16] yields that the dual of (5) is given by

$$\underset{\alpha}{\text{minimize}} \sum_{i,j=1}^{m} \sum_{y,y' \in \mathcal{Y}} \sum_{s,s' \in \mathcal{S}} \alpha_{iys} \alpha_{jy's'} K_{iys,jy's'} + \sum_{i=1}^{m} \sum_{y \in \mathcal{Y}} \sum_{s \in \mathcal{S}} \Delta(y_i, y, s) \alpha_{iys} \tag{6a}$$

$$\text{subject to } \lambda m \sum_{y \in \mathcal{Y}} \sum_{s \in \mathcal{S}} \alpha_{iys} = 1 \text{ for all } i \text{ and } \alpha_{iys} \geq 0. \tag{6b}$$

Here the entries of the kernel matrix $K$ are given by

$$K_{iys,jy's'} = \Gamma(y_i, y) \Gamma(y_j, y') \langle \phi(s(x_i), y) - \phi(s(x_i), y_i), \phi(s'(x_j), y') - \phi(s'(x_j), y_j) \rangle \tag{7}$$

This can be expanded into four kernel functions by using Eq. (4). Moreover, the connection between the dual coefficients $\alpha_{iys}$ and $f$ is given by

$$f(x', y') = \sum_{i=1}^{m} \sum_{y \in \mathcal{Y}} \sum_{s \in \mathcal{S}} \alpha_{iys} \left[ k((s(x_i), y), (x', y')) - k((s(x_i), y_i), (x', y')) \right]. \tag{8}$$

There are many strategies for attempting to minimize this regularized loss, either in the primal formulation or the dual, using either batch or online algorithms. In fact, a number of previous heuristics for dealing with invariances can be viewed as heuristics for approximately minimizing an approximation to an invariant loss similar to $l$. For this reason we believe a discussion of optimization is valuable *before* introducing specific applications of the invariance loss.

Whenever the are an unlimited combination of valid transformations and targets (i.e. the domain $\mathcal{S} \times \mathcal{Y}$ is infinite), the optimization above is a semi-infinite program, hence *exact* minimization of $R[f]$ or of its dual are essentially impossible. However, even is such cases it is possible to find approximate solutions efficiently by means of column generation. In the following we describe two algorithms exploiting this technique, which are valid for both infinite and finite programs. One based on a batch scenario, inspired by SVMStruct [16], and one based on an online setting, inspired by BMRM/Pegasos [15, 12].

## 3.1 A Variant of SVMStruct

The work of [16, 10] on SVMStruct-like optimization methods can be used directly to solve regularized risk minimization problems. The basic idea is to compute gradients of $l(x_i, y_i, f)$, either one observation at a time, or for the entire set of observations simultaneously and to perform updates in the dual space. While bundle methods work directly with gradients, solvers of the SVMStruct type are commonly formulated in terms of column generation on individual observations. We give an instance of SVMStruct for invariances in Algorithm 1. The basic idea is that instead of checking the constraints arising from the loss functions only for $y$ we check them for $(y, s)$, that is, an invariance in combination with a corresponding label which violates the margin most.

If we view the tuple $(s, y)$ as a "label" it is straightforward to see that the convergence results of [16] apply. That is, this algorithm converges to $\epsilon$ precision in $O(\epsilon^{-2})$ time. In fact, one may show, by solving the difference equation in the convergence proof of [16] that the rate can be improved to $O(\epsilon^{-1})$. We omit technical details here.

---

**Algorithm 1** SVMStruct for Invariances

---
1: **Input:** data $X$, labels $Y$, sample size $m$, tolerance $\epsilon$
2: Initialize $S_i = \emptyset$ for all $i$, and $w = 0$.
3: **repeat**
4:     **for** $i = 1$ **to** $m$ **do**
5:         $f(x', y') = \sum_i \sum_{(s,y) \in S_i} \alpha_{iz} \left[ k((s(x_i), y), (x', y')) - k((s(x_i), y_i), (x', y')) \right]$
6:         $(s^*, y^*) = \mathrm{argmax}_{s \in \mathcal{S}, y \in \mathcal{Y}} \Gamma(y_i, y)[f(s(x_i), y) - f(s(x_i), y_i)] + \Delta(y_i, y, s)$
7:         $\xi_i = \max(0, \max_{(s,y) \in S_i} \Gamma(y_i, y)[f(s(x_i), y) - f(s(x_i), y_i)] + \Delta(y_i, y, s))$
8:         **if** $\Gamma(y_i, y^*)[f(s^*(x_i), y^*) - f(s^*(x_i), y_i)] + \Delta(y_i, y^*, s^*) > \xi_i + \epsilon$ **then**
9:            Increase constraint set $S_i \leftarrow S_i \cup \{(s^*, y^*)\}$
10:           Optimize (6) using only $\alpha_{iz}$ where $z \in S_i$.
11:         **end if**
12:     **end for**
13: **until** $S$ has not changed in this iteration

---

## 3.2 An Application of Pegasos

Recently, Shalev-Shwartz et al. [12] proposed an online algorithm for learning optimization problems of type Eq. (5). Algorithm 2 is an adaptation of their method to learning with our convex invariance loss. In a nutshell, the algorithm performs stochastic gradient descent on the regularized version of the instantaneous loss while using a learning rate of $\frac{1}{\lambda t}$ and while projecting the current weight vector back to a feasible region $\|f\| \leq \sqrt{\frac{2R[0]}{\lambda}}$, should it exceed it.

---

**Algorithm 2** Pegasos for Invariances

---
1: **Input:** data $X$, labels $Y$, sample size $m$, iterations $T$,
2: Initialize $f_1 = 0$
3: **for** $t = 1$ **to** $T$ **do**
4:     Pick $(x, y) := (x_{t \bmod m}, y_{t \bmod m})$
5:     Compute constraint violator

$$(s^*, y^*) := \underset{\bar{s} \in \mathcal{S}, \bar{y} \in \mathcal{Y}}{\mathrm{argmax}} \; \Gamma(y, \bar{y}) \left[ f(\bar{s}(x), \bar{y}) - f(\bar{s}(x), y) \right] + \Delta(y, \bar{y}, \bar{s})$$

6:     Update $f_{t+1} = \left[ 1 - \frac{1}{t} \right] f_t + \frac{\Gamma(y, y^*)}{\lambda t} \left[ k((s^*(x), y), (\cdot, \cdot)) - k((s^*(x), y^*), (\cdot, \cdot)) \right]$
7:     **if** $\|f_{t+1}\| > \sqrt{\frac{2R[0]}{\lambda}}$ **then**
8:         Update $f_{t+t} \leftarrow \sqrt{\frac{2R[0]}{\lambda}} f_{t+1} / \|f_{t+1}\|$
9:     **end if**
10: **end for**

---

We can apply the convergence result from [12] directly to Algorithm 2. In this context note that the gradient with respect to $l$ is bounded by twice the norm of $\Gamma(y, y^*) [\phi(s(x), y^*) - \phi(s(x), y)]$, due to Eq. (3). We assume that the latter is given by $R$. We can apply [12, Lemma 1] immediately:

**Theorem 3** *Denote by $R_t[f] := l(x_{t \bmod m}, y_{t \bmod m}, f) + \frac{\lambda}{2} \|f\|^2$ the instantaneous risk at step $t$. In this case Algorithm 2 satisfies the following bound:*

$$\frac{1}{T} \sum_{t=1}^{T} R_t \left[ \frac{1}{T} \sum_{\bar{t}} f_{\bar{t}} \right] \leq \frac{1}{T} \sum_{t=1}^{T} R_t[f_t] \leq \min_{\|f\| \leq \sqrt{\frac{2R[0]}{\lambda}}} \frac{1}{T} \sum_{t=1}^{T} R_t[f] + \frac{R^2(1 + \log T)}{2\lambda T}. \tag{9}$$

In particular, if $T$ is a multiple of $m$ we obtain bounds for the regularized risk $R[f]$.

## 4 Related work and specific invariances

While the previous sections gave a theoretical description of the loss, we now discuss a number of special cases which can be viewed as instances of a convex invariance loss function presented here.

**Virtual Support Vectors (VSVs):** The most straightforward approach to incorporate prior knowledge is by adding "virtual" (data) points generated from existing dataset. An extension of this approach is to generate virtual points only from the support vectors (SVs) obtained from training on the original dataset [5]. The advantage of this approach is that it results in far fewer SV than training on all virtual points. However, it is not clear which objective it optimizes. Our current *loss based* approach does optimize an objective, and generates the required support vectors in the process of the optimization.

**Second Order Cone Programming for Missing and Uncertain Data:** In [2], the authors consider the case where the invariance is in the form of ellipsoids around the original point. This is shown to correspond to a second order cone program (SOCP). Instead of solving SOCP, we can solve an equivalent but unconstrained convex problem.

**Semidefinite Programming for Invariances:** Graepel and Herbrich [8] introduce a method for learning when the invariances are polynomial trajectories. They show that the problem is equivalent to an semidefinite program (SDP). Their formulation is again an instance of our general loss based approach. Since SDPs are typically hard to solve for large problems, it it is likely that the optimization scheme we suggest will perform considerably faster than standard SDP solvers.

**Robust Estimation:** Globerson and Roweis [7] address the case where invariances correspond to deletion of a subset of the features (i.e., setting their values to zero). This results in a quadratic program (QP) with a variables for each data point and feature in the training set. Solving such a large QP (e.g., $10^7$ variables for the MNIST dataset) is not practical, and again the algorithm presented here can be much more efficient. In fact, in the next section we introduce a generalization of the invariance in [7] and show how it can be optimized efficiently.

## 5 Experiments

Knowledge about invariances can be useful in a wide array of applications such as image recognition and document processing. Here we study two specific cases: handwritten digit recognition on the MNIST data, and spam filtering on the ECML06 dataset. Both examples are standard multiclass classification tasks, where $\Delta(y, y', s)$ is taken to be the 0/1 loss. Also, we take the margin scale $\Gamma(y, y')$ to be identically one. We used SVMStruct and BMRM as the solvers for the experiments.

### 5.1 Handwritten Digits Recognition

Humans can recognize handwritten digits even when they are altered in various ways. To test our invariant SVM (Invar-SVM) in this context, we used handwritten digits from the MNIST dataset [11] and modeled 20 invariance transformations: 1-pixel and 2-pixel shifts in 4 and 8 directions, rotations by $\pm 10$ degrees, scaling by $\pm 0.15$ unit, and shearing in vertical or horizontal axis by $\pm 0.15$ unit. To test the effect of learning with these invariances we used small training samples of $10, 20, \ldots, 50$ samples per digit. In this setting invariances are particularly important since they can compensate for the insufficient training data. We compared Invar-SVM to a related method where all possible transformations were applied in advance to each data point to create *virtual* samples. The virtual and original samples were used to train a multiclass SVM (VIR-SVM). Finally, we also trained a multiclass SVM that did not use any invariance information (STD-SVM). All of the aforementioned SVMs were trained using RBF kernel with well-chosen hyperparameters. For evaluation we used the standard MNIST test set.

Results for the three methods are shown in Figure 1. It can be seen that Invar-SVM and VIR-SVM, which use invariances, significantly improve the recognition accuracy compared to STD-SVM. This comes at a certain cost of using more support vectors, but for Invar-SVM the number of support vectors is roughly half of that in the VIR-SVM.

### 5.2 SPAM Filtering

The task of detecting spam emails is a challenging machine learning problem. One of the key difficulties with such data is that it can change over time as a result of attempts of spam authors to outwit spam filters [4]. In this context, the spam filter should be invariant to the ways in which a spam authors will change their style. One common mechanism of style alteration is the insertion of common words, and avoiding using specific keywords consistently over time. If documents are

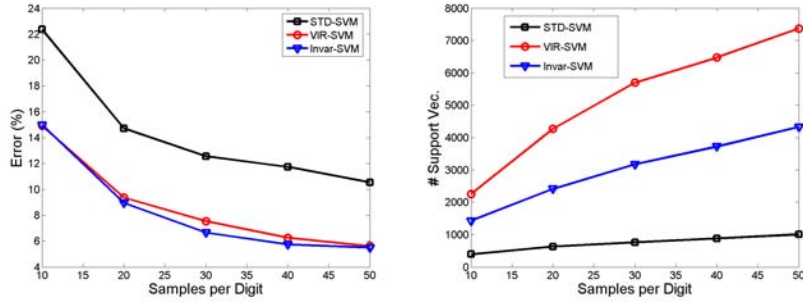

Figure 1: Results for the MNIST handwritten digits recognition task, comparing SVM trained on original samples (STD-SVM), SVM trained on original and virtual samples (VIR-SVM), and our convex invariance-loss method (Invar-SVM). Left figure shows the classification error as a function of the number of original samples per digit used in training. Right figure shows the number of support vectors corresponding to the optimum of each method.

represented using a bag-of-words, these two strategies correspond to incrementing the counts for some words, or setting it to zero [7].

Here we consider a somewhat more general invariance class (FSCALE) where word counts may be scaled by a maximum factor of $u$ (e.g., 1.5) and a minimum factor of $l$ (e.g., 0.5), and the maximum number of words subject to such perturbation is limited at $K$. Note that by setting $l = 0$ and $u = 1$ we specialize it to the feature deletion case (FDROP) in [7].

The invariances we consider are thus defined by

$$s(\mathbf{x}) = \{\mathbf{x} \circ \alpha : \alpha \in [l, u]^d, l \leq 1 \leq u, \#\{i : \alpha_i \neq 1\} \leq K\}, \tag{10}$$

where $\circ$ denotes element-wise product, $d$ is the number of features, and $\#\{\cdot\}$ denotes the cardinality of the set. The set $\mathcal{S}$ is large so exhaustive enumeration is intractable. However, the search for optimal perturbation $s^*$ is a linear program and can be computed efficiently by Algorithm 3 in $O(d \log d)$ time.

We evaluated the performance of our invariance loss FSCALE and its special case FDROP as well as the standard hinge loss on ECML'06 Discovery Challenge Task A dataset.[4] This dataset consists of two subsets, namely evaluation set (`ecml06a-eval`) and tuning set (`ecml06a-tune`). `ecml06a-eval` has 4000/7500 training/testing emails with dimensionality 206908, and `ecml06a-tune` has 4000/2500 training/testing emails with dimensionality 169620. We selected the best parameters for each methods on `ecml06a-tune` and used them for the training on `ecml06a-eval`. Results and parameter sets are shown in Table 1. We also performed McNemar's Tests and rejected the null hypothesis that there is no difference between hinge and FSCALE/FDROP with $p$-value $< 10^{-32}$.

---

**Algorithm 3** FSCALE loss

---

1: **Input:** datum $x$, label $y$, weight vector $w \in \mathbb{R}^d$, invariance-loss parameters $(K, l, u)$
2: Initialize $i := 1$, $j := d$
3: $B := y * w \circ x$
4: $I := \text{IndexSort(B)}$, such that $B(I)$ is in ascending order
5: **for** $k = 1$ **to** $K$ **do**
6:    **if** $B[I[i]] * (1 - u) > B[I[j]] * (1 - l)$ **then**
7:       $x[I[i]] := x[I[i]] * u$ **and** $i := i + 1$
8:    **else**
9:       $x[I[j]] := x[I[j]] * l$ **and** $j := j - 1$
10:    **end if**
11: **end for**

| Loss | Average Accuracy % | Average AUC % | Parameters $(\lambda, K, l, u)$ |
|---|---|---|---|
| Hinge | 74.75 | 83.63 | (0.005,-,-,-) |
| FDROP | 81.73 | 87.79 | (0.1,14,0,1) |
| FSCALE | 83.71 | 89.14 | (0.01,10,0.5,8) |

Table 1: SPAM filtering results on `ecml06a-eval` averaged over 3 testing subsets. $\lambda$ is regularization constant, $(K, l, u)$ are parameters for invariance-loss methods. The loss FSCALE and its special case FDROP statistically significantly outperform the standard hinge loss (Hinge).

## 6   Summary

We have presented a general approach for learning using knowledge about invariances. Our cost function is essentially a worst case margin loss, and thus its optimization only relies on finding the worst case invariance for a given data point and model. This approach can allow us to solve invariance problems which previously required solving very large optimization problems (e.g. a QP in [7]). We thus expect it to extend the scope of learning with invariances both in terms of the invariances used and efficiency of optimization.

**Acknowledgements:** We thank Carlos Guestin and Bob Williamson for fruitful discussions. Part of the work was done when CHT was visiting NEC Labs America. NICTA is funded through the Australian Government's *Backing Australia's Ability* initiative, in part through the ARC. This work was supported in part by the IST Programme of the European Community, under the PASCAL Network of Excellence, IST-2002-506778.

## Footnotes

[1]For more nontrivial examples see, e.g. [16, 14] and the references therein.

[2]Normally $\Delta = 0$ if $\bar{y}(x) = y$ but this is not strictly necessary.

[3]Such scaling has been shown to be extremely important and effective in many practical problems especially in structured prediction tasks. For example, the key difference between the large margin settings of [14] and [16] is the incorporation of a sequence-length dependent margin scaling.

[4]`http://www.ecmlpkdd2006.org/challenge.html`

## References

[1] Y. Abu-Mostafa. A method for learning from hints. In S. J. Hanson, J. D. Cowan, and C. L. Giles, editors, *NIPS 5*, 1992.

[2] C. Bhattacharyya, K. S. Pannagadatta, and A. J. Smola. A second order cone programming formulation for classifying missing data. In L. K. Saul, Y. Weiss, and L. Bottou, editors, *NIPS 17*, 2005.

[3] C. J. C. Burges. Geometry and invariance in kernel based methods. In B. Schölkopf, C. J. C. Burges, and A. J. Smola, editors, *Advances in Kernel Methods — Support Vector Learning*, pages 89–116, Cambridge, MA, 1999. MIT Press.

[4] N. Dalvi, P. Domingos, Mausam, S. Sanghai, and D. Verma. Adversarial classification. In *KDD*, 2004.

[5] D. DeCoste and B. Schölkopf. Training invariant support vector machines. *Machine Learning*, 46:161–190, 2002.

[6] M. Ferraro and T. M. Caelli. Lie transformation groups, integral transforms, and invariant pattern recognition. *Spatial Vision*, 8:33–44, 1994.

[7] A. Globerson and S. Roweis. Nightmare at test time: Robust learning by feature deletion. In *ICML*, 2006.

[8] T. Graepel and R. Herbrich. Invariant pattern recognition by semidefinite programming machines. In S. Thrun, L. Saul, and B. Schölkopf, editors, *NIPS 16*, 2004.

[9] G. E. Hinton. Learning translation invariant recognition in massively parallel networks. In *Proceedings Conference on Parallel Architectures and Laguages Europe*, pages 1–13. Springer, 1987.

[10] T. Joachims. Training linear SVMs in linear time. In *KDD*, 2006.

[11] Y. LeCun, L. D. Jackel, L. Bottou, A. Brunot, C. Cortes, J. S. Denker, H. Drucker, I. Guyon, U. A. Müller, E. Säckinger, P. Simard, and V. Vapnik. Comparison of learning algorithms for handwritten digit recognition. In F. Fogelman-Soulié and P. Gallinari, editors, *ICANN*, 1995.

[12] S. Shalev-Shwartz, Y. Singer, and N. Srebro. Pegasos: Primal estimated sub-gradient solver for SVM. In *ICML*, 2007.

[13] P. Simard, Y. LeCun, and J. Denker. Efficient pattern recognition using a new transformation distance. In S. J. Hanson, J. D. Cowan, and C. L. Giles, editors, *NIPS 5*, 1993.

[14] B. Taskar, C. Guestrin, and D. Koller. Max-margin Markov networks. In S. Thrun, L. Saul, and B. Schölkopf, editors, *NIPS 16*, 2004.

[15] C.H. Teo, Q. Le, A.J. Smola, and S.V.N. Vishwanathan. A scalable modular convex solver for regularized risk minimization. In *KDD*, 2007.

[16] I. Tsochantaridis, T. Joachims, T. Hofmann, and Y. Altun. Large margin methods for structured and interdependent output variables. *J. Mach. Learn. Res.*, 6:1453–1484, 2005.

